# Discovering Weakly-Interacting Factors in a Complex Stochastic Process

**Charlie Frogner**
School of Engineering and Applied Sciences
Harvard University
Cambridge, MA 02138
`frogner@seas.harvard.edu`

**Avi Pfeffer**
School of Engineering and Applied Sciences
Harvard University
Cambridge, MA 02138
`avi@eecs.harvard.edu`

## Abstract

Dynamic Bayesian networks are structured representations of stochastic processes. Despite their structure, exact inference in DBNs is generally intractable. One approach to approximate inference involves grouping the variables in the process into smaller factors and keeping independent beliefs over these factors. In this paper we present several techniques for decomposing a dynamic Bayesian network *automatically* to enable factored inference. We examine a number of features of a DBN that capture different types of dependencies that will cause error in factored inference. An empirical comparison shows that the most useful of these is a heuristic that estimates the mutual information introduced between factors by one step of belief propagation. In addition to features computed over entire factors, for efficiency we explored scores computed over pairs of variables. We present search methods that use these features, pairwise and not, to find a factorization, and we compare their results on several datasets. Automatic factorization extends the applicability of factored inference to large, complex models that are undesirable to factor by hand. Moreover, tests on real DBNs show that automatic factorization can achieve significantly lower error in some cases.

## 1 Introduction

Dynamic Bayesian networks (DBNs) are graphical model representations of discrete-time stochastic processes. DBNs generalize hidden Markov models and are used for modeling a wide range of dynamic processes, including gene expression [1] and speech recognition [2]. Although a DBN represents the process's transition model in a structured way, all variables in the model might become jointly dependent over the course of the process and so exact inference in a DBN usually requires tracking the full joint probability distribution over all variables; it is generally intractable. Factored inference approximates this joint distribution over all variables as the product of smaller distributions over groups of variables (factors) and in this way enables tractable inference for large, complex models. Inference algorithms based on this idea include Boyen-Koller [3], the Factored Frontier [4] and Factored Particle Filtering [5].

Factored inference has generally been demonstrated for models that are factored by hand. In this paper we will show that it is possible algorithmically to select a good factorization, thus not only extending the applicability of factored inference to larger models, for which it might be undesireable manually to choose a factorization, but also allowing for *better* (and sometimes 'non-obvious') factorizations. The quality of a factorization is defined by the amount of error incurred by repeatedly discarding the dependencies between factors and treating them as independent during inference. As such we formulate the goal of our algorithm as the minimization over factorizations of an objective that describes the error we expect due to this type of approximation. For this purpose we have examined a range of features that can be computed from the specification of the DBN, based both on

the underlying graph structure and on two essential conceptions of weak interaction between factors: the degree of separability [6] and mutual information. For each principle we investigated a number of heuristics. We find that the mutual information between factors that is introduced by one step of belief state propagation is especially well-suited to the problem of finding a good factorization.

Complexity is an issue in searching for good factors, as the search space is large and the scoring heuristics themselves are computationally intensive. We compare several search methods for finding factors that allow for different tradeoffs between the efficiency and the quality of the factorization. The fastest is a graph partitioning algorithm in which we find a $k$-way partition of a weighted graph with edge-weights being pairwise scores between variables. Agglomerative clustering and local search methods use the higher-order scores computed between whole factors, and are hence slower while finding better factorizations. The more expensive of these methods are most useful when run *offline*, for example when the DBN is to be used for online inference and one cares about finding a good factorization ahead of time. We additionally give empirical results on two other real DBN models as well as randomly-generated models. Our results show that dynamic Bayesian networks can be decomposed efficiently and automatically, enabling wider applicability of factored inference. Furthermore, tests on real DBNs show that using automatically found factors can in some cases yield significantly lower error than using factors found by hand.

## 2 Background

A *dynamic Bayesian network* (DBN), [7] [8], represents a dynamic system consisting of some set of variables that co-evolve in discrete timesteps. In this paper we are dealing with discrete variables. We denote the set of variables in the system by $\mathbf{X}$, with the *canonical variables* being those that directly influence at least one variable in the next timestep. We call the probability distribution over the possible states of the system at a given timestep the *belief state*. The DBN gives us the probabilities of transitioning from any given system state at $t$ to any other system state at time $t + 1$, and it does so in a factored way: the probability that a variable takes on a given state at $t + 1$ depends only on the states of a subset of the variables in the system at $t$. We can hence represent this transition model as a Bayesian network containing the variables in $\mathbf{X}$ at timestep $t$, denoted $\mathbf{X}_t$, and the variables in $\mathbf{X}$ at timestep $t + 1$, say $\mathbf{X}_{t+1}$ – this is called a *2-TBN* (for two-timeslice Bayesian network). By inferring the belief state over $\mathbf{X}_{t+1}$ from that over $\mathbf{X}_t$, and conditioning on observations, we propagate the belief state through the system dynamics to the next timestep. The specification of a DBN also includes a prior belief state at time $t = 0$.

Note that, although each variable at $t + 1$ may only depend on a small subset of the variables at $t$, its state might be correlated implicitly with the state of any variable in the system, as the influence of any variable might propagate through intervening variables over multiple timesteps. As a result, the whole belief state over $\mathbf{X}$ (at a given timestep) in general is not factored. Boyen and Koller, [3], find that, despite this fact, we can factor the system into components whose belief states are kept independently, and the error incurred by doing so remains bounded over the course of the process. The BK algorithm hence approximates the belief state at a given timestep as the product of the local belief states for the factors (their marginal distributions), and does exact inference to propagate this approximate belief state to the next timestep. Both the Factored Frontier, [4], and Factored Particle, [5], algorithms also rely on this idea of a factored belief state representation.

In [9] and [6], Pfeffer introduced conditions under which a single variable's (or factor's) marginal distribution will be propagated accurately through belief state propagation, in the BK algorithm. The *degree of separability* is a property of a conditional probability distribution that describes the degree to which that distribution can be decomposed as the sum of simpler conditional distributions, each of which depends on only a subset of the conditioning variables. For example, let $\mathbf{p}(Z|XY)$ give the probability distribution for $Z$ given $X$ and $Y$. If $\mathbf{p}(Z|XY)$ is separable in terms of $X$ and $Y$ to a degree $\alpha$, this means that we can write

$$\mathbf{p}(Z|XY) = \alpha[\gamma \mathbf{p}_X(Z|X) + (1 - \gamma)\mathbf{p}_Y(Z|Y)] + (1 - \alpha)\mathbf{p}_{XY}(Z|XY) \qquad (1)$$

for some conditional probability distributions $\mathbf{p}_X(Z|X)$, $\mathbf{p}_X(Z|Y)$, and $\mathbf{p}_{XY}(Z|XY)$ and some parameter $\gamma$. We will say that the degree of separability is the maximum $\alpha$ such that there exist $\mathbf{p}_X(Z|X)$, $\mathbf{p}_X(Z|Y)$, and $\mathbf{p}_{XY}(Z|XY)$ and $\gamma$ that satisfy (1). [9] and [6] have shown that if a system is highly separable then the BK algorithm produces low error in the components' marginal distributions.

Previous work has explored bounds on the error encountered by the BK algorithm. [3] showed that the error over the course of a process is bounded with respect to the error incurred by repeatedly projecting the exact distribution onto the factors as well as the *mixing rate* of the system, which can be thought of as the rate at which the stochasticity of the system causes old errors to be forgotten. [10] analyzed the error introduced between the exact distribution and the factored distribution by just one step of belief propagation. The authors noted that this error can be decomposed as the sum of conditional mutual information terms between variables in different factors and showed that each such term is bounded with respect to the mixing rate of the subsystem comprising the variables in that term. Computing the value of this error decomposition, unfortunately, requires one to examine a distribution over all of the variables in the model, which can be intractable. Along with other heuristics, we examined two approaches to automatic factorization that seek directly to exploit the above results, labeled *in-degree* and *out-degree* in Table 1.

## 3   Automatic factorization with pairwise scores

We first investigated a collection of features, computable from the specification of the DBN, that capture different types of pairwise dependencies between variables. These features are based both on the 2-TBN graph structure and on two conceptions of interaction: the degree of separability and mutual information. These methods allow us to factorize a DBN without computing expensive whole-factor scores.

### 3.1   Algorithm: Recursive min-cut

We use the following algorithm to find a factorization using only scores between pairs of variables. We build an undirected graph over the canonical variables in the DBN, weighting each edge between two variables with their pairwise score. An obvious algorithm for finding a partition that minimizes pairwise interactions between variables in different factors would be to compute a $k$-way min-cut, taking, say, the best-scoring such partition in which all factors are below a size limit. Unfortunately, on larger models this approach underperforms, yielding many partitions of size one. Instead we find that a good factorization can be achieved by computing a recursive min-cut, recurring until all factors are smaller than the pre-defined maximum size. We begin with all variables in a single factor. As long as there exists a factor whose weight is larger than the maximum, we do the following. For each factor that is too large, we search over the number of smaller factors, $k$, into which to divide the large factor, for each $k$ computing the $k$-way min-cut factorization of the variables in the large factor. In our experiments we use a spectral graph partitioning algorithm, [11], e.g. We choose the $k$ that minimizes the overall sum of between-factor scores. This is repeated until all factors are of sizes less than the maximum. This min-cut approach is designed only to use scores computed between pairs of variables, and so it sacrifices optimality for significant speed gains.

### 3.2   Pairwise scores

*Graph structure*

As a baseline in terms of speed and simplicity, we first investigated three types of pairwise graph relationships between variables that are indicative of different types of dependency.

- *Children of common parents.* Suppose that two variables at time $t + 1$, $X_{t+1}$ and $Y_{t+1}$, depend on some common parents $\mathbf{Z}_t$. As $X$ and $Y$ share a common, direct influence, we might expect them to to become correlated over the course of the process. The score between $X$ and $Y$ is the number of parents they share in the 2-TBN.

- *Parents of common children.* Suppose that $X_t$ and $Y_t$ jointly influence common children $\mathbf{Z}_{t+1}$. Then we might *care* more about any correlations between $X$ and $Y$, because they jointly influence $\mathbf{Z}$. If $X$ and $Y$ are placed in separate factors, then the accuracy of $\mathbf{Z}$'s marginal distribution will depend on how correlated $X$ and $Y$ were. Here the score between $X$ and $Y$ is the number of children they share in the 2-TBN.

- *Parent to child.* If $Y_{t+1}$ directly depends on $X_t$, or $X_{t+1}$ on $Y_t$, then we expect them to be correlated. The score between $X$ and $Y$ is the number of edges between them in the 2-TBN.

*Degree of separability*

The degree of separability for a given factor's conditional distribution in terms of the other factors gives a measure of how accurately the belief state for that factor will be propagated via that conditional distribution to the next timestep, in BK inference. When a factor's conditional distribution is highly separable in terms of the other factors, ignored dependencies between the other factors lead to relatively small errors in that factor's marginal belief state after propagation. We can hence use the degree of separability as an objective to be maximized: we want to find the factorization that yields the highest degree of separability for each factor's conditional distribution. Computing the degree of separability is a constrained optimization problem, and [12] gives an approximate method of solution. For distributions over many variables the degree of separability is quite expensive to compute, as the number of variables in the optimization grows exponentially with the number of discrete variables in the input conditional distribution. Computing the degree of separability for a small distribution is, however, reasonably efficient. In adapting the degree of separability to a pairwise score for the min-cut algorithm, we took two approaches.

- *Separability of the pair's joint conditional distribution*: We assign a score to the pair of canonical variables $X$ and $Y$ equal to the degree of separability for the joint conditional distribution $\mathbf{p}(X_{t+1}Y_{t+1}|Parents(X_{t+1}) \cup Parents(Y_{t+1}))$. We want to *maximize* this value for variables that are joined in a factor, as a high degree of separability implies that the error of the factor marginal distribution after propagation in BK will be low. Note that the degree of separability is defined in terms of *groups* of parent variables. If we have, for example, $\mathbf{p}(Z|WXY)$, then this distribution might be highly separable in terms of the groups $XY$ and $W$, but not in terms of $WX$ and $Y$. If, however, $\mathbf{p}(Z|WXY)$ is highly separable in terms of $W$, $X$ and $Y$ grouped separately, then it is at least as separable in terms of any other groupings. We compute the degree of separability for the above joint conditional distribution in terms of the parents taken separately.

- *Non-separability between parents of a common child*: If two parents are highly *non*-separable in a common child's conditional distribution, then the child's marginal distribution can be rendered inaccurate by placing these two parents in different components. For two variables $X$ and $Y$, we refer to the shared children of $X_t$ and $Y_t$ in timeslice $t+1$ as $\mathbf{Z}_{t+1}$. The strength of interaction between $X$ and $Y$ is defined to be the average degree of non-separability for each variable in $\mathbf{Z}_{t+1}$ in terms of its parents taken separately. The degree of non-separability is one minus the degree of separability.

*Mutual information*

Whereas the degree of separability is a property of a single factor's conditional distribution, the mutual information between two factors measures their joint dependencies. To compute it exactly requires, however, that we obtain a joint distribution over the two factors. All we are given is a DBN defining the conditional distribution over the next timeslice given the previous, and some initial distribution over the variables at time 1. In order to obtain a suitable joint distribution over the variables at $t+1$ we must assume a prior distribution over the variables at time $t$. We therefore examine several features based on the mutual information that we can compute from the DBN in this way, to capture different types of dependencies.

- *Mutual information after one timestep*: We assume a prior distribution over the variables at time $t$ and do one step of propagation to get a marginal distribution over $X_{t+1}$ and $Y_{t+1}$. We then use this marginal to compute the mutual information between $X$ and $Y$, thus estimating the degree of dependency between $\mathbf{X}$ and $\mathbf{Y}$ that results from one step of the process.

- *Mutual information between timeslices $t$ and $t+1$*: We measure the dependencies resulting from $\mathbf{X}$ and $\mathbf{Y}$ directly influencing each other between timeslices: the more information $\mathbf{X}_t$ carries about $\mathbf{Y}_{t+1}$, the more we expect them to become correlated as the process evolves. Again, we assume a prior distribution at time $t$ and use this to obtain the joint distribution $\mathbf{p}(\mathbf{Y}_{t+1}\mathbf{X}_t))$, from which we can calculate their mutual information. We sum the mutual information between $X_t$ and $Y_{t+1}$ and that between $Y_t$ and $X_{t+1}$ to get the score.

- *Mutual information from the joint over both timeslices*: We take into account all possible direct influences between $\mathbf{X}$ and $\mathbf{Y}$, by computing the mutual information between the sets of variables $(\mathbf{X}_t \cup \mathbf{X}_{t+1})$ and $(\mathbf{Y}_t \cup \mathbf{Y}_{t+1})$. As before, we assume a prior distribution at time $t$ to compute a joint distribution $\mathbf{p}((\mathbf{X}_t \cup \mathbf{X}_{t+1}) \cup (\mathbf{Y}_t \cup \mathbf{Y}_{t+1}))$, from which we can get the mutual information.

There are many possibilities for a prior distribution at time $t$. We can assume a uniform distribution, in which case the resulting mutual information values are exactly those introduced by one step of inference, as all variables are independent at time $t$. More costly would be to generate samples from the DBN and to do inference, computing the average mutual information values observed over the steps of inference. We found that, on small examples, there was little practical benefit to doing the latter. For simplicity we use the uniform prior, although the effects of different prior assumptions deserves further inquiry.

## 3.3  Empirical comparison

We compared the preceding pairwise scores by factoring randomly-generated DBNs, using the BK algorithm for belief state monitoring. We computed two error measures. The first is the joint belief state error, which is the relative entropy between the product of the factor marginal belief states and the exact joint belief state. The second is the average factor belief state error, which is the average over all factors of the relative entropy between each factor's marginal distribution and the equivalent marginal distribution from the exact joint belief state. We were constrained in choosing datasets on which exact inference is tractable, which limited both the number of state variables and the number of parameters per variable. Note that in our tables the joint KL distance is always given in terms of $10^{-2}$, while the factor marginal KL distance is in terms of $10^{-4}$.

For this comparison we used two datasets. The first is a large, relatively uncomplicated dataset that is intended to elucidate basic distinctions between the different heuristics. It consists of 400 DBNs, each of which contains 12 binary-valued state variables and 4 noisy observation variables. We tried to capture the tendency in real DBNs for variables to depend on a varying number of parents by drawing the number of parents for each variable from a gaussian distribution of mean 2 and standard deviation 1 (rounding the result and truncating at zero), and choosing parents uniformly from among the other variables. In real models variables usually, but not always, depend on themselves in the previous timeslice, and each variable in our networks also depended on itself with a probability of 0.75. Finally, the parameters for each variable were drawn randomly with a uniform prior.

The second dataset is intended to capture more complicated structures commonly seen in real DBNs: determinisim and context-specific independence. It consists of 50 larger models, each with 20 binary state variables and 8 noisy observation variables. Parents and parameters were chosen as before, except that in this case we chose several variables to be deterministic, each computing a boolean function of its parents, and several other variables to have tree-structured context-specific independence. To generate context-specific independence, the variable's parents were randomly permuted and between one half and all of the parents were chosen each to induce independence between the child variable and the parents lower in the tree, conditional upon one of its states.

The results are shown in Table 1. For reference we have shown two additional methods that minimize the maximum out-degree and in-degree of factors. These are suggested by Boyen and Koller as a means of controlling the mixing rate of factored inference, which is used to bound the error. In all cases, the mutual-information based factorizations, and in particular the mutual information after one timestep, yielded lower error, both in the joint belief state and in the factor marginal belief states. The degree of separability is apparently not well-adapted to a pairwise score, given that it is naturally defined in terms of an entire factor.

## 4  Exploiting higher-order interactions

The pairwise heuristics described above do not take into account higher-order properties of whole groups of variables: the mutual information between two factors is usually not exactly the sum of its constituent pairwise information relationships, and the degree of separability is naturally formulated in terms of a whole factor's conditional distribution and not between arbitrary pairs of variables. Two search algorithms allow us to use scores computed for whole factors, and to find better factors while sacrificing speed.

### 4.1  Algorithms: Agglomerative clustering and local search

Agglomerative clustering begins with all canonical variables in separate factors, and at each step chooses a pair of factors to merge such that the score of the factorization is minimized. If a merger leads to a factor of size greater than some given maximum, it is ignored. The algorithm stops when no advantageous merger is found. As the factors being scored are always of relatively small size, agglomerative clustering allows us to use full-factor scores.

Table 1: Random DBNs with pairwise scores

| | 12 nodes | | 20 nodes/determinism/CSI | |
| --- | --- | --- | --- | --- |
| | Joint KL | Factor KL | Joint KL | Factor KL |
| | $\times 10^{-4}$ | $\times 10^{-2}$ | $\times 10^{-4}$ | $\times 10^{-2}$ |
| Out-degree | 2.50 | 1.25 | 16.0 | 10.0 |
| In-degree | 2.44 | 1.20 | 15.1 | 8.54 |
| Children of common parents | 2.61 | 1.87 | 15.5 | 10.0 |
| Parents of common children | 1.98 | 1.01 | 11.9 | 5.92 |
| Parent to child | 2.28 | 1.19 | 14.9 | 6.62 |
| Separability between parents | 2.69 | 1.09 | 15.3 | 14.0 |
| Separability of pairs of variables | 2.80 | 1.27 | 18.5 | 12.0 |
| Mut. information after timestep | **1.11** | **0.408** | **7.11** | **3.44** |
| Mut. information between timeslices | 1.62 | 0.664 | 9.73 | 4.96 |
| Mut. information from both timeslices | 1.65 | 0.575 | 10.5 | 5.15 |

Local search begins with some initial factorization and attempts to find a factorization of minimum score by iteratively modifying this factorization. More specifically, from any given factorization moves of the following three types are considered: create a new factor with a single node, move a single node from one factor into another, or swap a pair of nodes in different factor. At each iteration only those moves that do not yield a factor of size greater than some given maximum are considered. The move that yields the lowest score at that iteration is chosen. If there is no move that decreases the score (and so we have hit a local minimum), however, the factors are randomly re-initialized and the algorithm continues searching, terminating after a fixed number of iterations. The factorization with the lowest score of all that were examined is returned. As with agglomerative clustering, local search enables the use of full-factor scores. We have found that good results are achieved when the factors are initialized (and re-initialized) to be as large as possible. In addition, although the third type of move (swapping) is a composition of the other two, we have found that the sequence of moves leading to an advantageous swap is not always a path of strictly decreasing scores, and performance degrades without it.

We note that all of the algorithms benefit greatly from caching the components of the scores that are computed.

## 4.2 Empirical comparison

We verified that the results for the pairwise scores extend to whole-factor scores on a dataset of 120 randomly-generated DBNs, each of which contained 8 binary-valued state variables. We were significantly constrained in our choice of models by the complexity of computing the degree of separability for large distributions: even on these smaller models, doing agglomerative clustering with the degree of separability sometimes took over 2 hours and local search much longer. We have therefore confined our comparison to agglomerative clustering on 8-variable models. We divided the dataset into three groups to explore the effects of both extensive determinism and context-specific independence separately.

The mutual information after one timestep again produced the lowest error in both in the factor marginal belief states and in the joint belief state. For the networks with large amounts of context-specific independence, the degree of separability was always close to one, and this might have hampered its effectiveness for clustering. Interestingly, we see that agglomerative clustering can sometimes produce results that are worse than those for graph partitioning, although local search consistently outperforms the two. This may be due to the fact that agglomerative clustering tends to produce smaller clusters than the divisive approach. Finally, we note that, although determinism greatly increased error, the relative performance of the different heuristics and algorithms was unchanged. Local search consistently found lower-error factorizations.

We further compared the different algorithms on the dataset with 12 state variables per DBN, from Section 3.3, using the mutual information after one timestep score. It is perhaps surprising that the graph min-cut algorithm can perform comparably with the others, given that it is restricted to pairwise scores.

Table 2: Random DBNs using pairwise and whole-factor scores

| Score type/Search algorithm | 8 nodes | | 8 nodes/determ. | | 8 nodes/CSI | |
|---|---|---|---|---|---|---|
| | Joint | Factor | Joint | Factor | Joint | Factor |
| Separability between parents: | | | | | | |
|   Min-cut | 2.36 | 2.54 | 38.9 | 70 | 0.82 | 0.45 |
| Separability b/t pairs of variables: | | | | | | |
|   Min-cut | 2.42 | 2.12 | 27.2 | 139 | 0.56 | 0.31 |
| Whole-factor separability: | | | | | | |
|   Agglomerative | 2.19 | 1.23 | 31.1 | 61 | 0.99 | 0.46 |
| Mut. info. after one timestep: | | | | | | |
|   Min-cut | 1.20 | 1.00 | 18.1 | 44 | 0.25 | 0.11 |
|   Agglomerative | 1.15 | 1.13 | 19.0 | 43 | 0.20 | 0.11 |
|   Local search | **1.05** | **0.90** | **13.8** | 32 | **0.18** | **0.098** |
| Mut. info. between timeslices: | | | | | | |
|   Min-cut | 1.62 | 1.17 | 27.7 | 47 | 0.55 | 0.24 |
|   Agglomerative | 1.60 | 1.45 | 27.6 | 61 | 0.53 | 0.32 |
|   Local search | 1.40 | 1.20 | 23.8 | 44 | 0.52 | 0.32 |
| Mut. info. both timeslices: | | | | | | |
|   Min-cut | 1.88 | 1.51 | 22.9 | 45 | 0.64 | 0.36 |
|   Agglomerative | 1.86 | 1.08 | 25.1 | 62 | 0.66 | 0.34 |
|   Local search | 1.70 | 0.95 | 23.1 | **26** | 0.58 | 0.29 |

## 5 Factoring real models

Boyen and Koller, [3], demonstrated factored inference on two models that were factored by hand: the Bayesian Automated Taxi network and the water network. Table 3 shows the performance of automatic factorization on these two DBNs. In both cases automatic factorization recovered reasonable factorizations that performed better than those found manually.

The Bayesian Automated Taxi (BAT) network, [13], is intended to monitor highway traffic and car state for an automated driving system. The DBN contains 10 persistent state variables and 10 observation variables. Local search with factors of 5 or fewer variables yielded exactly the 5+5 clustering given in the paper. When allowing 4 or fewer variables per factor, local search and agglomerative search both recovered the factorization (**[LeftClr], [RightClr], [LatAct+Xdot+InLane], [FwdAct+Ydot+Stopped+EngStatus], [FrontBackStatus]**), while graph min-cut found (**[EngStatus], [FrontBackStatus], [InLane], [Ydot], [FwdAct+Ydot+Stopped+EngStatus], [LatAct+LeftClr]**). The manual factorization from [3] is (**[LeftClr+RightClr+LatAct], [Xdot+InLane], [FwdAct+Ydot+Stopped+EngStatus], [FrontBackStatus]**). The error results are shown in Table 3. Local search took about 300 seconds to complete, while agglomerative clustering took 138 seconds and graph min-cut 12 seconds.

The water network is used for monitoring the biological processes of a water purification plant. It has 8 state variables and 4 observation variables (labeled A through H), and all variables are discrete with 3 or 4 states. The agglomerative and local search algorithms yielded the same result (**[A+B+C+E], [D+F+G+H]**) and graph min-cut was only slightly different (**[A+C+E], [D+F+G+H], [B]**). The manual factorization from [3] is (**[A+B],[C+D+E+F],[G+H]**). The results in terms of KL distance are shown in Figure 3. The automatically recovered factorizations were on average at least an order of magnitude better. Local search took about one minute to complete, while agglomerative clustering took 30 seconds and graph min-cut 3 seconds.

## 6 Conclusion

We compared several heuristics and search algorithms for automatically factorizing a dynamic Bayesian network. These techniques attempt to minimize an objective score that captures the extent to which dependencies that are ignored by the factored approximation will lead to error. The heuristics we examined are based both on the structure of the 2-TBN and on the concepts of degree of separability and mutual information. The mutual information after one step of belief propaga-

Table 3: Algorithm performance

| | 12-var. random | | BAT | | Water | |
|---|---|---|---|---|---|---|
| | Jnt. | Fact. | Jnt. | Fact. | Jnt. | Fact. |
| Min-cut | 1.08 | 0.433 | 14.7 | 0.723 | 0.430 | 1.32 |
| Agglomerative | 1.10 | 0.55 | 0.390 | 0.0485 | 0.0702 | 0.566 |
| Local search | 1.06 | 0.52 | 0.390 | 0.0485 | 0.0702 | 0.566 |
| Manual | - | - | 5.62 | 0.0754 | 3.12 | 2.12 |

tion has generally been greatly more effective than the others as an objective for factorization. We presented three search methods that allow for tradeoffs between computational complexity and the quality of the factorizations they produce. Recursive min-cut efficiently uses scores between pairs of variables, while agglomerative clustering and local search both use scores computed between whole factors – the latter two are slower, while achieving better results. Automatic factorization can extend the applicability of factored inference to larger models for which it is undesireable to find factors manually. In addition, tests run on real DBNs show that automatically factorized DBNs can achieve significantly lower error than hand-factored models. Future work might explore extensions to overlapping factors, which have been found to yield lower error in some cases.

**Acknowledgments**

This work was funded by an ONR project, with special thanks to Dr. Wendy Martinez.

# References

[1] Sun Yong Kim, Seiya Imot, and Satoru Miyano. Inferring gene networks from time series microarray data using dynamic Bayesian networks. *Briefings in Bioinformatics*, 2003.

[2] Geoffrey Zweig and Stuart Russell. Dynamic Bayesian networks for speech recognition. In *National Conference on Artificial Intelligence (AAAI)*, 1998.

[3] Xavier Boyen and Daphne Koller. Tractable inference for complex stochastic processes. In *Neural Information Processing Systems*, 1998.

[4] Kevin Murphy and Yair Weiss. The factored frontier algorithm for approximate inference in DBNs. In *Uncertainty in Artificial Intelligence*, 2001.

[5] Brenda Ng, Leonid Peshkin, and Avi Pfeffer. Factored particles for scalable monitoring. In *Uncertainty in Artificial Intelligence*, 2002.

[6] Avi Pfeffer. Approximate separability for weak interaction in dynamic systems. In *Uncertainty in Artificial Intelligence*, 2006.

[7] Thomas Dean and Keiji Kanazawa. A model for reasoning about persistence and causation. *Computational Intelligence*, 1989.

[8] Kevin Murphy. *Dynamic Bayesian networks: representation, inference and learning*. PhD thesis, U.C. Berkeley, Computer Science Division, 2002.

[9] Avi Pfeffer. Sufficiency, separability and temporal probabilistic models. In *Uncertainty in Artificial Intelligence*, 2001.

[10] Xavier Boyen and Daphne Koller. Exploiting the architecture of dynamic systems. In *Proceedings AAAI-99*, 1999.

[11] Andrew Ng, Michael Jordan, and Yair Weiss. On spectral clustering: analysis and an algorithm. In *Neural Information Processing Systems*, 2001.

[12] Charlie Frogner and Avi Pfeffer. Heuristics for automatically decomposing a dynamic Bayesian network for factored inference. Technical Report TR-04-07, Harvard University, 2007.

[13] Jeff Forbes, Tim Huang, Keiji Kanazawa, and Stuart Russell. The BATmobile: towards a Bayesian automatic taxi. In *International Joint Conference on Artificial Intelligence*, 1995.

